# Facial Expression Transfer with Input-Output Temporal Restricted Boltzmann Machines

Matthew D. Zeiler[1], Graham W. Taylor[1], Leonid Sigal[2], Iain Matthews[2], and Rob Fergus[1]

[1]Department of Computer Science, New York University, New York, NY 10012
[2]Disney Research, Pittsburgh, PA 15213

## Abstract

We present a type of Temporal Restricted Boltzmann Machine that defines a probability distribution over an output sequence conditional on an input sequence. It shares the desirable properties of RBMs: efficient exact inference, an exponentially more expressive latent state than HMMs, and the ability to model nonlinear structure and dynamics. We apply our model to a challenging real-world graphics problem: facial expression transfer. Our results demonstrate improved performance over several baselines modeling high-dimensional 2D and 3D data.

## 1 Introduction

Modeling temporal dependence is an important consideration in many learning problems. One can capture temporal structure either explicitly in the model architecture, or implicitly through latent variables which can act as a "memory". Feedforward neural networks which incorporate fixed delays into their architecture are an example of the former. A limitation of these models is that temporal context is fixed by the architecture instead of inferred from the data. To address this shortcoming, recurrent neural networks incorporate connections between the latent variables at different time steps. This enables them to capture arbitrary dynamics, yet they are more difficult to train [2].

Another family of dynamical models that has received much attention are probabilistic models such as Hidden Markov Models and more general Dynamic Bayes nets. Due to their statistical structure, they are perhaps more interpretable than their neural-network counterparts. Such models can be separated into two classes [19]: tractable models, which permit an exact and efficient procedure for inferring the posterior distribution over latent variables, and intractable models which require approximate inference. Tractable models such as Linear Dynamical Systems and HMMs are widely applied and well understood. However, they are limited in the types of structure that they can capture. These limitations are exactly what permit simple exact inference. Intractable models, such as Switching LDS, Factorial HMMs, and other more complex variants of DBNs permit more complex regularities to be learned from data. This comes at the cost of using approximate inference schemes, for example, Gibbs sampling or variational inference, which introduce either a computational burden or poorly approximate the true posterior.

In this paper we focus on Temporal Restricted Boltzmann Machines [19,20], a family of models that permits tractable inference but allows much more complicated structure to be extracted from time series data. Models of this class have a number of attractive properties: 1) They employ a *distributed* state space where multiple factors interact to explain the data; 2) They permit nonlinear dynamics and multimodal predictions; and 3) Although maximum likelihood is intractable for these models, there exists a simple and efficient approximate learning algorithm that works well in practice.

We concentrate on modeling the distribution of an output sequence conditional on an input sequence. Recurrent neural networks address this problem, though in a non-probabilistic sense. The Input-Output HMM [3] extends HMMs by conditioning both their dynamics and emission model on an input sequence. However, the IOHMM is representationally limited by its simple discrete state in

the same way as a HMM. Therefore we extend TRBMs to cope with input-output sequence pairs. Given the conditional nature of a TRBM (its hidden states and observations are conditioned on short histories of these variables), conditioning on an external input is a natural extension to this model.

Several real-world problems involve sequence-to-sequence mappings. This includes motion-style transfer [9], economic forecasting with external indicators [13], and various tasks in natural language processing [6]. Sequence classification is a special case of this setting, where a scalar target is conditioned on an input sequence. In this paper, we consider facial expression transfer, a well-known problem in computer graphics. Current methods considered by the graphics community are typically linear (e.g., methods based on blendshape mapping) and they do not take into account dynamical aspects of the facial motion itself. This makes it difficult to retarget the facial articulations involved in speech. We propose a model that can encode a complex nonlinear mapping from the motion of one individual to another which captures facial geometry and dynamics of both source and target.

## 2  Related work

In this section we discuss several latent variable models which can map an input sequence to an output sequence. We also briefly review our application field: facial expression transfer.

### 2.1  Temporal models

Among probabilistic models, the Input-Output HMM [3] is most similar to the architecture we propose. Like the HMM, the IOHMM is a generative model of sequences but it models the distribution of an output sequence conditional on an input, while the HMM simply models the distribution of an output sequence. The IOHMM is also trained with a more discriminative-style EM-based learning paradigm than HMMs. A similarity between IOHMMs and TRBMs is that in both models, the dynamics and emission distributions are formulated as neural networks. However, the IOHMM state space is a multinomial while TRBMs have binary latent states. A $K$-state TRBM can thus represent the history of a time series using $2^K$ state configurations while IOHMMs are restricted to $K$ settings.

The Continuous Profile Model [12] is a rich and robust extension of dynamic time warping that can be applied to many time series in parallel. The CPM has a discrete state-space and requires an input sequence. Therefore it is a type of conditional HMM. However, unlike the IOHMM and our proposed model, the input is unobserved, making learning completely unsupervised.

Our approach is also related to the many proposed techniques for supervised learning with structured outputs. The problem of simultaneously predicting multiple, correlated variables has received a great deal of recent attention [1]. Many of these models, including the one we propose, are formally defined as undirected graphs whose potential functions are functions of some input. In Graph Transformer Networks [11] the dependency structure on the outputs is chosen to be sequential, which decouples the graph into pairwise potentials. Conditional Random Fields [10] are a special case of this model with linear potential functions. These models are trained discriminatively, typically with gradient descent, where our model is trained generatively using an approximate algorithm.

### 2.2  Facial expression transfer

Facial expression transfer, also called motion retargeting or cross-mapping, is the act of adapting the motion of an actor to a target character. It, as well as the related fields of facial performance capture and performance-driven animation, have been very active research areas over the last several years.

According to a review by Pighin [15], the two most important considerations for this task are facial model parameterization (called "the rig" in the graphics industry) and the nature of the chosen cross-mapping. A popular parameterization is "blendshapes" where a rig is a set of linearly combined facial expressions each controlled by a scalar weight. Retargeting amounts to estimating a set of blending weights at each frame of the source data that accurately reconstructs the target frame. There are many different ways of selecting blendshapes, from simply selecting a set of sufficient frames from the data, to creating models based on principal components analysis. Another common parameterization is to simply represent the face by its vertex, polygon or spline geometry. The downside of this approach is that this representation has many more degrees of freedom than are present in an actual facial expression.

A linear function is the most common choice for cross-mapping. While it is simple to estimate from data, it cannot produce subtle nonlinear motion required for realistic graphics applications. An

example of this approach is [5] which uses a parametric model based on eigen-points to reliably synthesize simple facial expressions but ultimately fails to capture more subtle details. Vlasic et al. [23] have proposed a multilinear mapping where variation in appearance across the source and target is explicitly separated from the variation in facial expression. None of these models explicitly incorporate dynamics into the mapping, which is a limitation addressed by our approach.

Finally, we note that Susskind et al. [18] have used RBMs for facial expression generation, but not retargeting. Their work is focused on static rather than temporal data.

## 3  Modeling dynamics with Temporal Restricted Boltzmann Machines

In this section we review the Temporal Restricted Boltzmann Machine. We then introduce the Input-Output Temporal Restricted Boltzmann Machine which extends the architecture to model an output sequence conditional on an input sequence.

### 3.1  Temporal Restricted Boltzmann Machines

A Restricted Boltzmann Machine [17] is a bipartite Markov Random Field consisting of a layer of stochastic observed variables ("visible units") connected to a layer of stochastic latent variables ("hidden units"). The absence of connections between hidden units ensures they are conditionally independent given a setting of the visible units, and vice-versa. This simplifies inference and learning.

The RBM can be extended to model temporal data by conditioning its visible units and/or hidden units on a short history of their activations. This model is called a Temporal Restricted Boltzmann Machine [19]. Conditioning the model on the previous settings of the hidden units complicates inference. Although one can approximate the posterior distribution with the filtering distribution (treating the past setting of the hidden units as fixed), we choose to use a simplified form of the model which conditions only on previous *visible* states [20]. This model inherits the most important computational properties of the standard RBM: simple, exact inference and efficient approximate learning.

RBMs typically have binary observed variables and binary latent variables but to model real-valued data (e.g., the parameterization of a face), we can use a modified form of the TRBM with conditionally independent linear-Gaussian observed variables [7]. The model, depicted in Fig. 1 (left), defines a joint probability distribution over a real-valued representation of the current frame of data, $\mathbf{v}_t$, and a collection of binary latent variables, $\mathbf{h}_t, h_j \in \{0, 1\}$:

$$p(\mathbf{v}_t, \mathbf{h}_t | \mathbf{v}_{<t}) = \exp\left(-E(\mathbf{v}_t, \mathbf{h}_t | \mathbf{v}_{<t})\right) / Z(\mathbf{v}_{<t}). \tag{1}$$

For notational simplicity, we concatenate a short history of data at $t-1, \ldots, t-N$ into a vector which we call $\mathbf{v}_{<t}$. The distribution specified by Eq. 1 is conditional on this history and normalized by a quantity $Z$ which is intractable to compute exactly[1] but not needed for inference nor learning.

The joint distribution is characterized by an "energy function":

$$E(\mathbf{v}_t, \mathbf{h}_t | \mathbf{v}_{<t}) = \sum_i \frac{1}{2}(v_{i,t} - \hat{a}_{i,t})^2 - \sum_j h_{j,t}\hat{b}_{j,t} - \sum_{ij} W_{ij}v_{i,t}h_{j,t} \tag{2}$$

which captures pairwise interactions between variables, assigning high energy to improbable configurations and low energy to probable configurations. In the first term, each visible unit contributes a quadratic penalty that depends on its deviation from a "dynamic mean" determined by the history:

$$\hat{a}_{i,t} = a_i + \sum_k A_{ki}\mathbf{v}_{k,<t} \tag{3}$$

where $k$ indexes the history vector. Weight matrix $A$ and offset vector $\mathbf{a}$ (with elements $a_i$) parameterize the autoregressive relationship between the history and current frame of data. Each hidden unit $h_j$ contributes a linear offset to the energy which is also a function of the history:

$$\hat{b}_{j,t} = b_j + \sum_k B_{kj}\mathbf{v}_{k,<t}. \tag{4}$$

Weight matrix $B$ and offset $\mathbf{b}$ (with elements $b_j$) parameterize the relationship between the history and the latent variables. The final term of Eq. 2 is a bi-linear constraint on the interaction between the current setting of the visible units and hidden units, characterized by matrix $W$.

The density for observation $\mathbf{v}_t$ conditioned on the past can be expressed by marginalizing out the binary hidden units in Eq. 1:

$$p(\mathbf{v}_t|\mathbf{v}_{<t}) = \sum_{\mathbf{h}_t} p(\mathbf{v}_t, \mathbf{h}_t|\mathbf{v}_{<t}) = \sum_{\mathbf{h}_t} \exp\left(-E(\mathbf{v}_t, \mathbf{h}_t|\mathbf{v}_{<t})\right)/Z(\mathbf{v}_{<t}), \qquad (5)$$

while the probability of observing a *sequence*, $\mathbf{v}_{(N+1):T}$, given an $N$-frame history $\mathbf{v}_{1:N}$, is simply the product of all the local conditional probabilities up to time $T$, the length of a sequence:

$$p(\mathbf{v}_{(N+1):T}|\mathbf{v}_{1:N}) = \prod_{t=N+1}^{T} p(\mathbf{v}_t|\mathbf{v}_{<t}). \qquad (6)$$

The TRBM has been used to generate and denoise sequences [19, 20], as well as a prior in multi-view person tracking [22]. In all cases, it requires an initialization, $\mathbf{v}_{1:N}$, to perform these tasks. Alternatively, by learning a prior model of $\mathbf{v}_{1:N}$ it could easily extended to model sequences non-conditionally, i.e., defining $p(\mathbf{v}_{1:T})$.

### 3.2 Input-Output Temporal Restricted Boltzmann Machines

Ultimately we are interested in learning a probabilistic mapping from an input sequence, $\mathbf{s}_{1:T}$ to an output sequence, $\mathbf{v}_{1:T}$. In other words, we seek a model that defines $p(\mathbf{v}_{1:T}|\mathbf{s}_{1:T})$. However, the TRBM only defines a distribution over an output sequence $p(\mathbf{v}_{1:T})$. Extending this model to learn an input-output mapping is the primary contribution of this paper. Without loss of generality, we will assume that in addition to having access to the complete history of the input, we also have access to the first $N$ frames of the output. Therefore we seek to model $p(\mathbf{v}_{(N+1):T}|\mathbf{v}_{1:N}, \mathbf{s}_{1:T})$. By placing an $N^{\text{th}}$ order Markov assumption on the current output, $\mathbf{v}_t$, that is, assuming conditional independence on all other variables given an $N$-frame history of $\mathbf{v}_t$ and an $N+1$-frame history of the input (up to and including time $t$), we can operate in an online setting:

$$p(\mathbf{v}_{(N+1):T}|\mathbf{v}_{1:N}, \mathbf{s}_{1:T}) = \prod_{t=N+1}^{T} p(\mathbf{v}_t|\mathbf{v}_{<t}, \mathbf{s}_{<=t}). \qquad (7)$$

where we have used the shorthand $\mathbf{s}_{<=t}$ to describe a vector that concatenates a window over the input at time $t, t-1, \ldots, t-N$. Note that in an offline setting, it is simple to generalize the model by conditioning the term inside the product on an arbitrary window of the source (which may include source observations past time $t$).

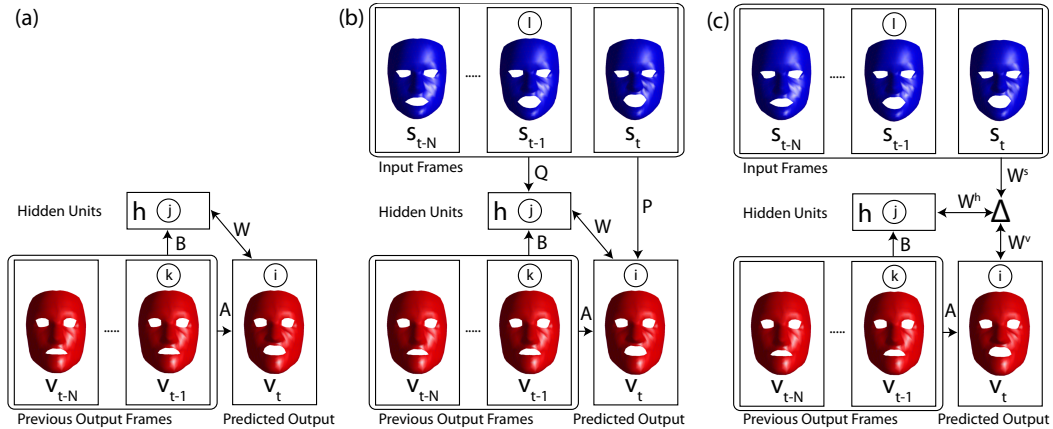

Figure 1: Left: A Temporal Restricted Boltzmann Machine. Middle: An Input-Output Temporal Restricted Boltzmann Machine. Right: A factored third-order IOTRBM (FIOTRBM).

We can easily adapt the TRBM to model $p(\mathbf{v}_t|\mathbf{v}_{<t}, \mathbf{s}_{<=t})$ by modifying its energy function to incorporate the input. The general form of energy function remains the same as Eq. 2 but it is now also conditioned on $\mathbf{s}_{<=t}$ by redefining the dynamic biases (Eq. 3 and 4) as follows:

$$\hat{a}_{it} = a_i + \sum_k A_{ki}\mathbf{v}_{k,<t} + \sum_l P_{li}\mathbf{s}_{l,<=t} \tag{8}$$

$$\hat{b}_{jt} = b_j + \sum_k B_{kj}\mathbf{v}_{k,<t} + \sum_l Q_{lj}\mathbf{s}_{l,<=t} \tag{9}$$

where $l$ is an index over elements of the input vector. Therefore the matrix P ties the input linearly to the output (much like existing simple models) but the matrix Q also allows the input to nonlinearly interact with the output through the latent variables $\mathbf{h}$. We call this model an Input-Output Temporal Restricted Boltzmann Machine (IOTRBM). It is depicted in Fig. 1 (middle).

A desirable criterion for training the model is to maximize the conditional log likelihood of the data:

$$\mathcal{L} = \sum_{t=N+1}^{T} \log p(\mathbf{v}_t|\mathbf{v}_{<t}, \mathbf{s}_{<=t}). \tag{10}$$

However, the gradient of Eq. 10 with respect to the model parameters $\theta = \{W, A, B, P, Q, \mathbf{a}, \mathbf{b}\}$ is difficult to compute analytically due to the normalization constant $Z$. Therefore, Contrastive Divergence (CD) learning is typically used in place of maximum likelihood. It follows the approximate gradient of an objective function that is the difference between two Kullback-Leibler divergences [8]. It is widely used in practice and tends to produce good generative models [4].

The CD updates for the IOTRBM have a common form (see the supplementary material for details):

$$\Delta\theta_i \propto \sum_{t=N+1}^{T} \left\langle \frac{\partial E(\mathbf{v}_t, \mathbf{h}_t|\mathbf{v}_{<t}, \mathbf{s}_{<=t})}{\partial\theta_i} \right\rangle_{\text{data}} - \left\langle \frac{\partial E(\mathbf{v}_t, \mathbf{h}_t|\mathbf{v}_{<t}, \mathbf{s}_{<=t})}{\partial\theta_i} \right\rangle_{\text{recon}} \tag{11}$$

where $\langle\cdot\rangle_{\text{data}}$ is an expectation with respect to the training data distribution, and $\langle\cdot\rangle_{\text{recon}}$ is the $M$-step reconstruction distribution as obtained by alternating Gibbs sampling, starting with the visible units clamped to the training data. The input and output history stay fixed during Gibbs sampling. CD requires two main operations: 1) sampling the latent variables, given a window of the input and output,

$$p(h_{j,t} = 1|\mathbf{v}_t, \mathbf{v}_{<t}, \mathbf{s}_{<=t}) = \left(1 + \exp(-\sum_i W_{ij}v_{i,t} - \hat{b}_{jt})\right)^{-1}, \tag{12}$$

and 2) reconstructing the output data, given the latent variables:

$$v_{i,t}|\mathbf{h}_t, \mathbf{v}_{<t}, \mathbf{s}_{<=t} \sim \mathcal{N}\left(v_{it}; \sum_j W_{ij}h_{j,t} + \hat{a}_{i,t}, 1\right). \tag{13}$$

Eq. 12 and 13 are alternated $M$ times to arrive at the $M$-step quantities used in the weight updates. More details are given in Sec. 4.

### 3.3 Factored Third-order Input-Output Temporal Restricted Boltzmann Machines

In an IOTRBM the input and target history can only modify the hidden units and current output through additive biases. There has been recent interest in exploring higher-order RBMs in which variables interact multiplicatively [14, 16, 21]. Fig. 1 (right) shows an IOTRBM whose parameters $W, Q$ and $P$ have been replaced by a three-way weight tensor defining a multiplicative interaction between the three sets of variables. The introduction of the tensor results in the number of model parameters becoming cubic and therefore we factor the tensor into three matrices: $W^{\mathbf{s}}, W^{\mathbf{h}}$, and $W^{\mathbf{v}}$. These parameters connect the input, hidden units, and current target, respectively to a set of deterministic units which modulate the connections between variables. The introduction of these factors corresponds to a kind of low-rank approximation to the original interaction tensor, that uses $O(K^2)$ parameters instead of $O(K^3)$.

The energy function of this model is:

$$E(\mathbf{v}_t, \mathbf{h}_t | \mathbf{v}_{<t}, \mathbf{s}_{<=t}) = \sum_i \frac{1}{2}(v_{i,t} - \hat{a}_{i,t})^2 - \sum_j h_{j,t}\hat{b}_{j,t} - \sum_f \sum_{ijl} W_{if}^{\mathbf{v}} W_{jf}^{\mathbf{h}} W_{lf}^{\mathbf{s}} v_{i,t} h_{j,t} s_{l,<=t} \quad (14)$$

where $f$ indexes factors and $\hat{a}_{i,t}$ and $\hat{b}_{j,t}$ are defined by Eq. 3 and 4 respectively. Weight updates all have the same form as Eq. 11 (see the supplementary material for details). The conditional distribution of the latent variables given the other variables becomes,

$$p(h_{j,t} = 1 | \mathbf{v}_t, \mathbf{v}_{<t}, \mathbf{s}_{<=t}) = \left(1 + \exp(-\sum_f W_{jf}^{\mathbf{h}} \sum_i W_{if}^{\mathbf{v}} v_{i,t} \sum_l W_{lf}^{\mathbf{s}} s_{l,<=t} - \hat{b}_{jt})\right)^{-1} \quad (15)$$

and the reconstruction distribution becomes,

$$v_{i,t} | \mathbf{h}_t, \mathbf{v}_{<t}, \mathbf{s}_{<=t} \sim \mathcal{N}\left(v_{it}; \sum_f W_{if}^{\mathbf{v}} \sum_j W_{jf}^{\mathbf{h}} h_{j,t} \sum_l W_{lf}^{\mathbf{s}} s_{l,<=t} + \hat{a}_{i,t}, 1\right). \quad (16)$$

## 4 Experiments

We evaluate the IOTRBM on two facial expression transfer datasets, one based on 2D motion capture and the other on 3D motion capture. On both datasets we compare our model against three baselines:

**Linear regression (LR)**: We perform a regularized linear regression between each frame of the input to each frame of the output. The model is solved analytically by least squares. The regularization parameter is set by cross-validation on the training set.

$N$**th-order Autoregressive[2] model (AR)**: This model improves on linear regression by also considering linear dynamics through the history of the input and output. Again through regularized least squares we fit a matrix that maps from a concatenation of the $(N + 1)$-frame input window $\mathbf{s}_{<=t}$ and $N$-frame target window, $\mathbf{v}_{<t}$.

**Multilayer perceptron**: A nonlinear model with one deterministic hidden layer, the same cardinality as the IOTRBM. The input is the concatenation of the source and target history, the output is the current target frame. We train with a nonlinear conjugate gradient method.

These baselines were chosen to highlight the main difference of our approach over the majority of techniques proposed for this application, namely the consideration of dynamics and the use of a nonlinear mapping through latent variables. We also tried an IORBM, that is, an IOTRBM with no target history. It consistently performed worse than the IOTRBM, and we do not report its results.

**Details of learning** All models saw a window of 4 input frames (3 previous + 1 current) and 6 previous output frames, with the exception of linear regression which only saw the current input. For the IOTRBM models, we found that initializing the parameters $A$ and $P$ to the solution found by the autoregressive model gave slightly better results. All other parameters were initialized to small random values. For CD learning we set the learning rates for $A$ and $P$ to $10^{-6}$ and for all other parameters to $10^{-3}$. This was done to prevent strong correlations from dominating early in learning. All parameters used a fixed weight decay of $0.02$ and momentum of $0.75$. As suggested by [21], we added a small amount of Gaussian noise ($\sigma = 0.1$) to the output history to make the model more robust to unseen outputs during prediction (recall that the model sees true outputs at training time, but is fed back predictions at test time).

### 4.1 2D facial expression transfer

The first dataset we consider consists of facial motion capture of two subjects who were asked to speak the same phrases. It has 186 trials, totaling 10414 fames per subject. Each frame is 180 dimensional, representing the $x$ and $y$ position of 90 facial markers. Each pair of sequences has been manually time-aligned based on a phonetic transcription so they are synchronized between subjects.

| | RMS Marker Error (mm) | | | | | | |
|---|---|---|---|---|---|---|---|
| Split<br>Model | S1 | S2 | S3 | S4 | S5 | S6 | Mean |
| Linear regression | 6.19 | 6.18 | 6.19 | 5.85 | 6.13 | 6.34 | 6.15 ± 0.15 |
| Autoregressive | 5.43 | **5.22** | 5.67 | 5.37 | 5.37 | 5.76 | 5.47 ± 0.20 |
| MLP | 5.30 | 5.28 | 5.76 | 5.31 | 5.28 | 5.31 | 5.37 ± 0.19 |
| IOTRBM | **5.31** | 5.27 | **5.71** | **5.14** | **5.17** | **5.08** | **5.28 ± 0.22** |
| FIOTRBM | 5.41 | 5.43 | 5.76 | 5.42 | 5.45 | 5.46 | 5.49 ± 0.13 |

Table 1: 2D dataset. Mean RMS error on test output sequences.

| | Input noise | | | Output noise | | | Input & Output Noise | | |
|---|---|---|---|---|---|---|---|---|---|
| Noise<br>Model | 0.01 | 0.1 | 1 | 0.01 | 0.1 | 1 | 0.01 | 0.1 | 1 |
| Linear regression | 6.48 | 15.05 | 136.2 | N/A | | | | | |
| Autoregressive | 5.83 | 10.48 | 84.40 | 5.78 | 7.24 | 36.19 | 5.85 | 11.26 | 94.35 |
| MLP | 5.40 | 5.42 | 6.80 | 5.40 | 5.43 | 6.37 | 5.40 | 5.43 | 7.55 |
| IOTRBM | **5.06** | **5.07** | **5.39** | **5.07** | **5.18** | 8.48 | **5.07** | **5.17** | 8.57 |
| FIOTRBM | 5.46 | 5.46 | 5.66 | 5.46 | 5.46 | **5.56** | 5.46 | 5.46 | **5.82** |

Table 2: 2D dataset. Mean RMS error (in mm) under noisy input and output history (Split 6).

**Preprocessing** We found the original data to exhibit significant random relative motion between the two faces throughout the entire sequences which could not reasonably be modeled. Therefore, we transformed the data with an affine transform on all markers in each frame such that a select few nose and skull markers per frame (stationary facial locations) were approximately fixed relative to the first frame of the source sequences. Both the input and output were reduced to 30 dimensions by retaining only their first 30 principal components. This maintained 99.9% of the variance in the data. Finally, the data was normalized to have zero mean and scaled by the average standard deviation of all the elements in the training set.

We evaluate the various methods on 6 random arbitrary splits of the dataset. In each case, 150 complete sequences are maintained for training and the remaining 36 sequences are used for testing. Each model is presented with the first 6 frames of the true test output and successive 4-frame windows of the true test input. The exception is the linear regression model, which only sees the current input. Therefore prediction is measured from the $7^{\text{th}}$ frame onward.

The IOTRBM produces its final output by initializing its visible units with the current previous frame plus a small amount of Gaussian noise and then performing 30 alternating Gibbs steps. At the last step, we do not sample the hidden units. This predicted output frame now becomes the most recent frame in the output history and we iterate forward. The results show a IOTRBM with 30 hidden units. We also tried a model with 100 hidden units which performed slightly worse. Finally, we include the performance of a factored, third-order IOTRBM. This model used 30 hidden units and 50 factors.

We report RMS marker error in mm where the mean is taken over all markers, frames and test sequences (Table 1). Not surprisingly, the IOTRBM consistently outperforms linear regression. In all but two splits (where performance is comparable) the IOTRBM outperforms the AR model. Mean performance over the splits shows an advantage to our approach. This is also qualitatively apparent in videos we have attached as supplementary material that superimpose the true target with predictions from the model. We encourage the reader to view the attached videos, as certain aesthetic properties such as the tradeoff between smoothness and responsiveness are not captured by RMS error. We observed that on the 2D dataset, the FIOTRBM had no advantage over the simpler IOTRBM.

To compare the robustness of each model to corrupted inputs or outputs, we added various amounts of white Gaussian noise to the input window, output history initialization or both during retargeting with a trained model. This is performed for data split S6 (though we observed similar results for other splits). The performance of each model is given in Table 2. The IOTRBM generally outperforms the baseline models in the presence of noise. This is most apparent in the case of input noise: the scenario we would most likely find in practice. However, under low to moderate output noise, we note that the IOTRBM is robust, to the point that it does not even require a valid $N$ frame output initialization to produce a sensible retargeting. Interestingly, we also observe the FIOTRBM performing well under high-noise conditions.

| | RMS Marker Error (mm) | | | | | |
|---|---|---|---|---|---|---|
| Split | S1 | S2 | S3 | S4 | S5 | Mean |
| Autoregressive | 2.12 | 2.98 | 2.44 | 2.26 | 2.46 | $2.45 \pm 0.33$ |
| MLP | 1.98 | 1.58 | 1.69 | 1.51 | **1.39** | $1.63 \pm 0.22$ |
| IOTRBM | 1.98 | 2.62 | 2.37 | 2.11 | 2.27 | $2.27 \pm 0.25$ |
| FIOTRBM | **1.70** | **1.54** | **1.55** | **1.42** | 1.48 | $\mathbf{1.54 \pm 0.10}$ |

Table 3: 3D dataset. Mean RMS error on test output sequences.

## 4.2   3D facial expression transfer

The second dataset we consider consists of facial motion capture data of two subjects, asked to perform a set of isolated facial movements based on FACS. The movements are more exaggerated than the speech performed in the 2D set. The dataset consists of two trials, totaling 1050 frames per subject. In contrast to the 2D set, the marker set used differs between subjects. The first subject has 313 markers (939 dimensions per frame) and the second subject has 332 markers (996 dimensions per frame). There is no correspondence between marker sets.

**Preprocessing** The 3D data was not spatially aligned. Both the input and output were PCA-reduced to 50 dimensions (99.9% of variance). We then normalized in the same way as for the 2D data.

We evaluate performance on 5 random splits of the 3D dataset, shown in Table 3. The IOTRBM and FIOTRBM models considered have identical architectures to the ones used for 2D data. We found empirically that increasing the noise level of the output history to $\sigma = 1$ improved generalization on the smaller dataset.

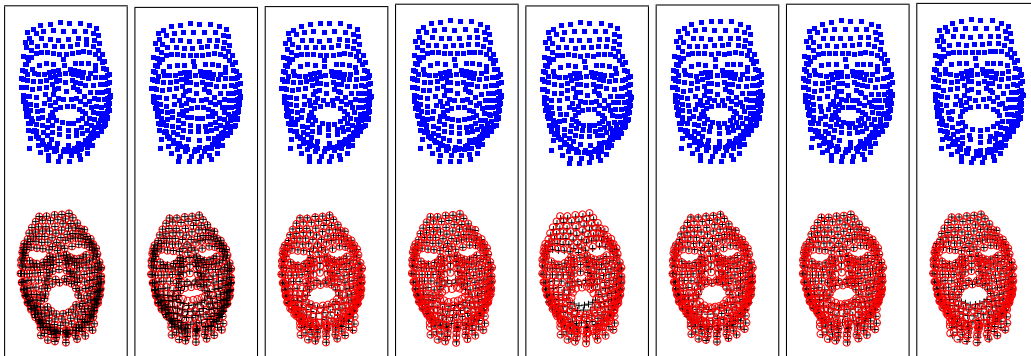

Figure 2: Retargeting with the third-order factored TRBM. We show every 30th frame. The top row shows the input. The bottom row shows the true target (circles) and the prediction from our model (crosses). This figure is best viewed in electronic form and zoomed.

Similar to the experiments with 2D data, the IOTRBM consistently outperforms the autoregressive model. However, it does not outperform the MLP. Interestingly, the factored, third-order model considerably improves on the performance of the standard IOTRBM and the MLP. Fig. 2 visualizes the predictions made by the FIOTRBM. We also refer the reader to videos included as supplementary material. These demonstrate a qualitative improvement of our models over the baselines considered.

## 5   Conclusion

We have introduced the Input-Output Temporal Restricted Boltzmann Machine, a probabilistic model for learning mappings between sequences. We presented two variants of the model, one with pairwise and one with third-order multiplicative interactions. Our experiments so far are limited to dynamic facial expression transfer, but nothing restricts the model to this domain.

Current methods for facial expression transfer are unable to factor out style in the retargeted motion, making it difficult to adjust the emotional content of the resulting facial animation. We are therefore interested in exploring extensions of our model that include style-based contextual variables (c.f., [21]).

**Acknowledgements**

The authors thank Rafael Tena and Sarah Hilder for assisting with data collection and annotation.

**Matlab code**

Code is available at: `http://www.matthewzeiler.com/pubs/nips2011/`.

## Footnotes

[1]To compute $Z$ exactly we would need to integrate over the joint space of all possible output configurations and all settings of the binary latent variables.

[2]This model considers the history of the source when predicting the target so it is not purely autoregressive.

## References

[1] G. H. Bakir, T. Hofmann, B. Schölkopf, A. J. Smola, B. Taskar, and S. V. N. Vishwanathan. *Predicting Structured Data*. MIT Press, 2007.

[2] Y. Bengio, P. Simard, and P. Frasconi. Learning long-term dependencies with gradient descent is difficult. *IEEE Transactions on Neural Networks*, 5(2):157–166, 1994.

[3] Y. Bengio and P. Frasconi. An input/output HMM architecture. In G. Tesauro, D. S. Touretzky, and T. K. Leen, editors, *Proc. NIPS 7*, pages 427–434, 1995.

[4] M. Carreira-Perpinan and G. Hinton. On contrastive divergence learning. In *AISTATS*, pages 59–66, 2005.

[5] E. Chuang and C. Bregler. Performance driven facial animation using blendshape interpolation. Technical report, Stanford University, 2002.

[6] R. Collobert and J. Weston. A unified architecture for natural language processing: deep neural networks with multitask learning. In *ICML*, pages 160–167, 2008.

[7] Y. Freund and D. Haussler. Unsupervised learning of distributions of binary vectors using 2-layer networks. In *Proc. NIPS 4*, 1992.

[8] G. Hinton. Training products of experts by minimizing contrastive divergence. *Neural Comput*, 14(8):1771–1800, 2002.

[9] E. Hsu, K. Pulli, and J. Popović. Style translation for human motion. *ACM Trans. Graph.*, 24(3):1082–1089, 2005.

[10] J. Lafferty, A. McCallum, and F. Pereira. Conditional random fields: Probabilistic models for segmenting and labeling sequence data. In *Proc. ICML*, pages 282–289, 2001.

[11] Y. LeCun, L. Bottou, Y. Bengio, and P. Haffner. Gradient-based learning applied to document recognition. *Proceedings of the IEEE*, 86(11):2278–2324, 1998.

[12] J. Listgarten, R. Neal, S. Roweis, and A. Emili. Multiple alignment of continuous time series. In *Proc. NIPS 17*, 2005.

[13] A. Mateo, A. Muñoz, and J. García-González. Modeling and forecasting electricity prices with input/output hidden Markov models. *IEEE Trans. on Power Systems*, 20(1):13–24, 1995.

[14] R. Memisevic and G. Hinton. Learning to represent spatial transformations with factored higher-order Boltzmann machines. *Neural Comput*, 22(6):1473–92, 2010.

[15] F. Pighin and J. P. Lewis. Facial motion retargeting. In *ACM SIGGRAPH 2006 Courses*, SIGGRAPH '06, New York, NY, USA, 2006. ACM.

[16] M. Ranzato and G. E. Hinton. Modeling pixel means and covariances using factorized Third-Order boltzmann machines. In *Proc. CVPR*, pages 2551–2558, 2010.

[17] P. Smolensky. Information processing in dynamical systems: Foundations of harmony theory. In D. E. Rumelhart, J. L. McClelland, et al., editors, *Parallel Distributed Processing: Volume 1: Foundations*, pages 194–281. MIT Press, Cambridge, MA, 1986.

[18] J. Susskind, G. Hinton, J. Movellan, and A. Anderson. Generating facial expressions with deep belief nets. In *Affective Computing, Focus on Emotion Expression, Synthesis and Recognition*. I-TECH Education and Publishing, 2008.

[19] I. Sutskever and G. Hinton. Learning multilevel distributed representations for high-dimensional sequences. In *Proc. AISTATS*, 2007.

[20] G. W. Taylor, G. E. Hinton, and S. Roweis. Modeling human motion using binary latent variables. In *Proc. NIPS 19*, 2007.

[21] G. Taylor and G. Hinton. Factored conditional restricted Boltzmann machines for modeling motion style. In *Proc. ICML*, pages 1025–1032, 2009.

[22] G. Taylor, L. Sigal, D. Fleet, and G. Hinton. Dynamical binary latent variable models for 3d human pose tracking. In *Proc. CVPR*, 2010.

[23] D. Vlasic, M. Brand, H. Pfister, and J. Popović. Face transfer with multilinear models. In *ACM SIGGRAPH 2005*, pages 426–433, 2005.

